# Global Versus Local Methods in Nonlinear Dimensionality Reduction

**Vin de Silva**
Department of Mathematics,
Stanford University,
Stanford. CA 94305
silva@math.stanford.edu

**Joshua B. Tenenbaum**
Department of Brain and Cognitive Sciences,
Massachusetts Institute of Technology,
Cambridge. MA 02139
jbt@ai.mit.edu

## Abstract

Recently proposed algorithms for nonlinear dimensionality reduction fall broadly into two categories which have different advantages and disadvantages: global (Isomap [1]), and local (Locally Linear Embedding [2], Laplacian Eigenmaps [3]). We present two variants of Isomap which combine the advantages of the global approach with what have previously been exclusive advantages of local methods: computational sparsity and the ability to invert conformal maps.

## 1 Introduction

In this paper we discuss the problem of nonlinear dimensionality reduction (NLDR): the task of recovering meaningful low-dimensional structures hidden in high-dimensional data. An example might be a set of pixel images of an individual's face observed under different pose and lighting conditions; the task is to identify the underlying variables (pose angles, direction of light, etc.) given only the high-dimensional pixel image data. In many cases of interest, the observed data are found to lie on an embedded submanifold of the high-dimensional space. The degrees of freedom along this submanifold correspond to the underlying variables. In this form, the NLDR problem is known as "manifold learning".

Classical techniques for manifold learning, such as principal components analysis (PCA) or multidimensional scaling (MDS), are designed to operate when the submanifold is embedded linearly, or almost linearly, in the observation space. More generally there is a wider class of techniques, involving iterative optimization procedures, by which unsatisfactory linear representations obtained by PCA or MDS may be "improved" towards more successful nonlinear representations of the data. These techniques include GTM [4], self organising maps [5] and others [6,7]. However, such algorithms often fail when nonlinear structure cannot simply be regarded as a perturbation from a linear approximation; as in the Swiss roll of Figure 3. In such cases, iterative approaches tend to get stuck at locally optimal solutions that may grossly misrepresent the true geometry of the situation.

Recently, several entirely new approaches have been devised to address this problem. These methods combine the advantages of PCA and MDS—computational efficiency; few free parameters; non-iterative global optimisation of a natural cost function—with the ability to recover the intrinsic geometric structure of a broad class of nonlinear data manifolds.

These algorithms come in two flavors: local and global. Local approaches (LLE [2], Laplacian Eigenmaps [3]) attempt to preserve the local geometry of the data; essentially, they seek to map nearby points on the manifold to nearby points in the low-dimensional representation. Global approaches (Isomap [1]) attempt to preserve geometry at all scales, mapping nearby points on the manifold to nearby points in low-dimensional space, and faraway points to faraway points.

The principal advantages of the global approach are that it tends to give a more faithful representation of the data's global structure, and that its metric-preserving properties are better understood theoretically. The local approaches have two principal advantages: (1) computational efficiency: they involve only sparse matrix computations which may yield a polynomial speedup; (2) representational capacity: they may give useful results on a broader range of manifolds, whose local geometry is close to Euclidean, but whose global geometry may not be.

In this paper we show how the global geometric approach, as implemented in Isomap, can be extended in both of these directions. The results are computational efficiency and representational capacity equal to or in excess of existing local approaches (LLE, Laplacian Eigenmaps), but with the greater stability and theoretical tractability of the global approach. Conformal Isomap (or C-Isomap) is an extension of Isomap which is capable of learning the structure of certain curved manifolds. This extension comes at the cost of making a uniform sampling assumption about the data. Landmark Isomap (or L-Isomap) is a technique for approximating a large global computation in Isomap by a much smaller set of calculations. Most of the work focuses on a small subset of the data, called the landmark points.

The remainder of the paper is in two sections. In Section 2, we describe a perspective on manifold learning in which C-Isomap appears as the natural generalisation of Isomap. In Section 3 we derive L-Isomap from a landmark version of classical MDS.

## 2   Isomap for conformal embeddings

### 2.1   Manifold learning and geometric invariants

We can view the problem of manifold learning as an attempt to invert a generative model for a set of observations. Let $Y$ be a $d$-dimensional domain contained in the Euclidean space $\mathbf{R}^d$, and let $f : Y \to \mathbf{R}^D$ be a smooth embedding, for some $D > d$. The object of manifold learning is to recover $Y$ and $f$ based on a given set $\{x_i\}$ of observed data in $\mathbf{R}^D$. The observed data arise as follows. Hidden data $\{y_i\}$ are generated randomly in $Y$, and are then mapped by $f$ to become the observed data, so $\{x_i = f(y_i)\}$.

The problem as stated is ill-posed: some restriction is needed on $f$ if we are to relate the observed geometry of the data to the structure of the hidden variables $\{y_i\}$ and $Y$ itself. We will discuss two possibilities. The first is that $f$ is an isometric embedding in the sense of Riemannian geometry; so $f$ preserves infinitesmal lengths and angles. The second possibility is that $f$ is a conformal embedding; it preserves angles but not lengths. Equivalently, at every point $y \in Y$ there is a scalar $s(y) > 0$ such that infinitesimal vectors at $y$ get magnified in length by a factor $s(y)$. The class of conformal embeddings includes all isometric embeddings as well as many other families of maps, including stereographic projections such as the Mercator projection.

One approach to solving a manifold learning problem is to identify which aspects of the geometry of $Y$ are invariant under the mapping $f$. For example, if $f$ is an isometric embedding then by definition infinitesimal distances are preserved. But more is true. The length of a path in $Y$ is defined by integrating the infinitesimal distance metric along the path. The same is true in $f(Y)$, so $f$ preserves path lengths. If $y, z$ are two points in $Y$, then the *shortest* path between $y$ and $z$ lying inside $Y$ is the same length as the shortest path

between $f(y)$ and $f(z)$ along $f(Y)$. Thus geodesic distances are preserved. The conclusion is that $Y$ is isometric with $f(Y)$, regarded as metric spaces under geodesic distance. Isomap exploits this idea by constructing the geodesic metric for $f(Y)$ approximately as a matrix, using the observed data alone.

To solve the conformal embedding problem, we need to identify an observable geometric invariant of conformal maps. Since conformal maps are locally isometric up to a scale factor $s(y)$, it is natural to try to estimate $s(y)$ at each point $f(y)$ in the observed data. By rescaling, we can then restore the original metric structure of the data and proceed as in Isomap. We can do this by noting that a conformal map $f$ rescales local volumes in $Y$ by a factor $s(y)^d$. Hence if the hidden data are sampled *uniformly* in $Y$, the local density of the observed data will be $1/s(y)^d$. It follows that the conformal factor $s(y)$ can be estimated in terms of the observed local data density, provided that the original sampling is uniform. C-Isomap implements a version of this idea which is independent of the dimension $d$.

This uniform sampling assumption may appear to be a severe restriction, but we believe it reflects a necessary tradeoff in dealing with a larger class of maps. Moreover, as we illustrate below, our algorithm appears in practice to be robust to moderate violations of this assumption.

## 2.2 The Isomap and C-Isomap algorithms

There are three stages to Isomap [1]:

1. Determine a *neighbourhood graph* $G$ of the observed data $\{x_i\}$ in a suitable way. For example, $G$ might contain $x_i x_j$ iff $x_j$ is one of the $k$ nearest neighbours of $x_i$ (and vice versa). Alternatively, $G$ might contain the edge $x_i x_j$ iff $|x_i - x_j| < \epsilon$, for some $\epsilon$.

2. Compute shortest paths in the graph for all pairs of data points. Each edge $x_i x_j$ in the graph is weighted by its Euclidean length $|x_i - x_j|$, or by some other useful metric.

3. Apply MDS to the resulting shortest-path distance matrix $D$ to find a new embedding of the data in Euclidean space, approximating $Y$.

The premise is that local metric information (in this case, lengths of edges $x_i x_j$ in the neighbourhood graph) is regarded as a trustworthy guide to the local metric structure in the original (latent) space. The shortest-paths computation then gives an estimate of the global metric structure, which can be fed into MDS to produce the required embedding.

It is known that Step 2 converges on the true geodesic structure of the manifold given sufficient data, and thus Isomap yields a faithful low-dimensional Euclidean embedding whenever the function $f$ is an isometry. More precisely, we have (see [8]):

**Theorem.** *Let $Y$ be sampled from a bounded convex region in $\mathbf{R}^d$, with respect to a density function $\alpha = \alpha(y)$. Let $f$ be a $C^2$-smooth isometric embedding of that region in $\mathbf{R}^D$. Given $\lambda, \mu > 0$, for a suitable choice of neighbourhood size parameter $\epsilon$ or $k$, we have*

$$1 - \lambda \leq \frac{\text{recovered distance}}{\text{original distance}} \leq 1 + \lambda$$

*with probability at least $1 - \mu$, provided that the sample size is sufficiently large.* [The formula is taken to hold for all pairs of points simultaneously.]

C-Isomap is a simple variation on Isomap. Specifically, we use the $k$-neighbours method in Step 1, and replace Step 2 with the following:

2a. Compute shortest paths in the graph for all pairs of data points. Each edge $x_i x_j$ in the graph is weighted by $|x_i - x_j| / \sqrt{M(i)M(j)}$. Here $M(i)$ is the mean distance of $x_i$ to its $k$ nearest neighbours.

Using similar arguments to those in [8], one can prove a convergence theorem for C-Isomap. The exact formula for the weights is not critical in the asymptotic analysis. The point is that the rescaling factor $\sqrt{M(i)M(j)}$ is an asymptotically accurate approximation to the conformal scaling factor in the neighbourhood of $x_i$ and $x_j$.

**Theorem.** *Let $Y$ be sampled uniformly from a bounded convex region in $\mathbf{R}^d$. Let $f$ be a $C^2$-smooth conformal embedding of that region in $\mathbf{R}^D$. Given $\lambda, \mu > 0$, for a suitable choice of neighbourhood size parameter $k$, we have*

$$1 - \lambda \leq \frac{\text{recovered distance}}{\text{original distance}} \leq 1 + \lambda$$

*with probability at least $1 - \mu$, provided that the sample size is sufficiently large.*

It is possible but unpleasant to find explicit lower bounds for the sample size. Qualitatively, we expect to require a larger sample size for C-Isomap since it depends on two approximations—local data density and geodesic distance—rather than one. In the special case where the conformal embedding is actually an isometry, it is therefore preferable to use Isomap rather than C-Isomap. This is borne out in practice.

## 2.3 Examples

We ran C-Isomap, Isomap, MDS and LLE on three "fishbowl" examples with different data distributions, as well as a more realistic simulated data set. We refer to Figure 1.

**Fishbowls:** These three datasets differ only in the probability density used to generate the points. For the *conformal fishbowl* (column 1), 2000 points were generated randomly uniformly in a circular disk $Y$ and then projected stereographically (hence conformally mapped) onto a sphere. Note the high concentration of points near the rim. There is no metrically faithful way of embedding a curved fishbowl inside a Euclidean plane, so classical MDS and Isomap cannot succeed. As predicted, C-Isomap does recover the original disk structure of $Y$ (as does LLE). Contrast with the *uniform fishbowl* (column 2), with data points sampled using a uniform measure on the fishbowl itself. In this situation C-Isomap behaves like Isomap, since the rescaling factor is approximately constant; hence it is unable to find a topologically faithful 2-dimensional representation. The *offset fishbowl* (column 3) is a perturbed version of the conformal fishbowl; points are sampled in $Y$ using a shallow Gaussian offset from center, then stereographically projected onto a sphere. Although the theoretical conditions for perfect recovery are not met, C-Isomap is robust enough to find a topologically correct embedding. LLE, in contrast, produces topological errors and metric distortion in both cases where the data are not uniformly sampled in $Y$ (columns 2 and 3).

**Face images:** Artificial images of a face were rendered as $128 \times 128$ pixel images and rasterized into 16384-dimensional vectors. The images varied randomly and independently in two parameters: left-right pose angle $\theta$ and distance from camera $y$. There is a natural family of conformal transformations for this data manifold, if we ignore perspective distortions in the closest images: namely $y \mapsto \lambda y$, for $\lambda > 0$, which has the effect of shrinking or magnifying the apparent size of images by a constant factor. Sampling uniformly in $\theta$ and in $\log y$ gives a data set approximately satisfying the required conditions for C-Isomap. We generated 2000 face images in this way, spanning the range indicated by Figure 2. All four algorithms returned a two-dimensional embedding of the data. As expected, C-Isomap returns the cleanest embedding, separating the two degrees of freedom reliably along the horizontal and vertical axes. Isomap returns an embedding which narrows predictably as the face gets further away. The LLE embedding is highly distorted.

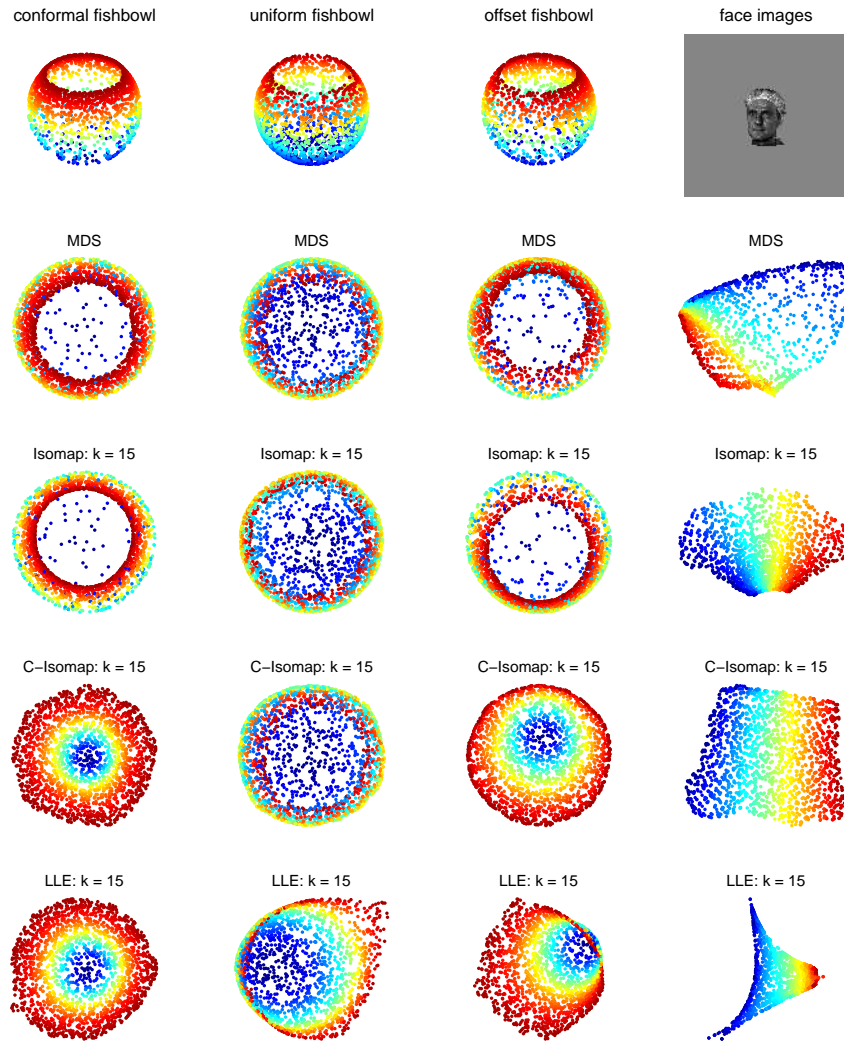

Figure 1: Four dimensionality reduction algorithms (MDS, Isomap, C-Isomap, and LLE) are applied to three versions of a toy "fishbowl" dataset, and to a more complex data manifold of face images.

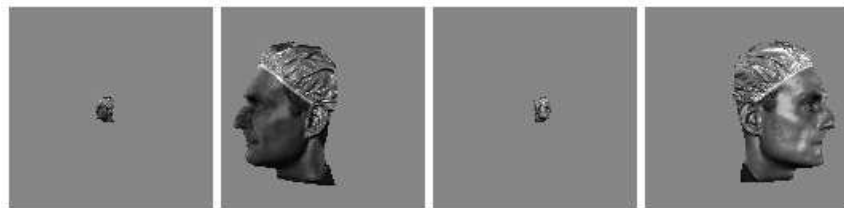

Figure 2: A set of 2000 face images were randomly generated, varying independently in two parameters: distance and left-right pose. The four extreme cases are shown.

## 3   Isomap with landmark points

The Isomap algorithm has two computational bottlenecks. The first is calculating the $N \times N$ shortest-paths distance matrix $D_N$. Using Floyd's algorithm this is $O(N^3)$; this can be improved to $O(kN^2 \log N)$ by implementing Dijkstra's algorithm with Fibonacci heaps ($k$ is the neighbourhood size). The second bottleneck is the MDS eigenvalue calculation, which involves a full $N \times N$ matrix and has complexity $O(N^3)$. In contrast, the eigenvalue computations in LLE and Laplacian Eigenmaps are sparse (hence considerably cheaper).

L-Isomap addresses both of these inefficiencies at once. We designate $n$ of the data points to be *landmark* points, where $n \ll N$. Instead of computing $D_N$, we compute the $n \times N$ matrix $D_{n,N}$ of distances from each data point to the landmark points only. Using a new procedure LMDS (Landmark MDS), we find a Euclidean embedding of the data using $D_{n,N}$ instead of $D_N$. This leads to an enormous savings when $n$ is much less than $N$, since $D_{n,N}$ can be computed using Dijkstra in $O(knN \log N)$ time, and LMDS runs in $O(n^2 N)$.

LMDS is feasible precisely because we expect the data to have a low-dimensional embedding. The first step is to apply classical MDS to the landmark points only, embedding them faithfully in $\mathbf{R}^\ell$. Each remaining point $x$ can now be located in $\mathbf{R}^\ell$ by using its known distances from the landmark points as constraints. This is analogous to the Global Positioning System technique of using a finite number of distance readings to identify a geographic location. If $n \geq \ell + 1$ and the landmarks are in general position, then there are enough constraints to locate $x$ uniquely. The landmark points may be chosen randomly, with $n$ taken to be sufficiently larger than the minimum $\ell + 1$ to ensure stability.

### 3.1   The Landmark MDS algorithm

LMDS begins by applying classical MDS [9,10] to the landmarks-only distance matrix $D_n$. We recall the procedure. The first step is to construct an "inner-product" matrix $B_n = -H_n \Delta_n H_n / 2$; here $\Delta_n$ is the matrix of squared distances and $H_n$ is the "centering" matrix defined by the formula $(H_n)_{ij} = \delta_{ij} - 1/n$. Next find the eigenvalues and eigenvectors of $B_n$. Write $\lambda_i$ for the positive eigenvalues (labelled so that $\lambda_1 \geq \lambda_2 \geq \ldots \geq \lambda_p$), and $\vec{v}_i$ for the corresponding eigenvectors (written as column vectors); non-positive eigenvalues are ignored. Then for $\ell \leq p$ the required optimal $\ell$-dimensional embedding vectors are given as the columns of the matrix:

$$L = \begin{bmatrix} \sqrt{\lambda_1} \cdot \vec{v}_1^T \\ \sqrt{\lambda_2} \cdot \vec{v}_2^T \\ \vdots \\ \sqrt{\lambda_\ell} \cdot \vec{v}_\ell^T \end{bmatrix}$$

The embedded data are automatically mean-centered with principal components aligned with the axes, most significant first. If $B_n$ has no negative eigenvalues, then the $p$-dimensional embedding is perfect; otherwise there is no exact Euclidean embedding.

The second stage of LMDS is to embed the remaining points in $\mathbf{R}^\ell$. Let $\Delta_x$ denote the column vector of squared distances between a data point $x$ and the landmark points. The embedding vector $\vec{x}$ is related linearly to $\Delta_x$ by the formula:

$$\vec{x} = \frac{1}{2} L^\sharp (\bar{\Delta}_n - \Delta_x)$$

where $\bar{\Delta}_n$ is the column mean of $\Delta_n$ and $L^\sharp$ is the pseudoinverse transpose of $L$:

$$L^\sharp = \begin{bmatrix} \vec{v}_1^T / \sqrt{\lambda_1} \\ \vec{v}_2^T / \sqrt{\lambda_2} \\ \vdots \\ \vec{v}_\ell^T / \sqrt{\lambda_\ell} \end{bmatrix}$$

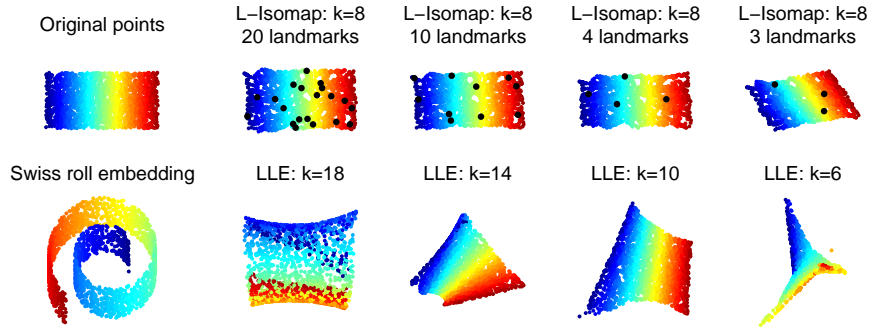

Figure 3: L-Isomap is stable over a wide range of values for the sparseness parameter $n$ (the number of landmarks). Results from LLE are shown for comparision.

The final (optional) stage is to use PCA to realign the data with the coordinate axes. A full discussion of LMDS will appear in [11]. We note two results here:

1. If $x$ is a landmark point, then the embedding given by LMDS is consistent with the original MDS embedding.

2. If the distance matrix $D_{n,N}$ can be represented exactly by a Euclidean configuration in $\mathbf{R}^\ell$, and if the landmarks are chosen so that their affine span in that configuration is $\ell$-dimensional (i.e. in general position), then LMDS will recover the configuration exactly, up to rotation and translation.

A good way to satisfy the affine span condition is to pick $\ell + 1$ landmarks randomly, plus a few extra for stability. This is important for Isomap, where the distances are inherently slightly noisy. The robustness of LMDS to noise depends on the matrix norm $\|L^\sharp\| = 1/\sqrt{\lambda_\ell}$. If $\lambda_\ell$ is very small, then all the landmarks lie close to a hyperplane and LMDS performs poorly with noisy data. In practice, choosing a few extra landmark points gives satisfactory results.

## 3.2   Example

Figure 3, shows the results of testing L-Isomap on a Swiss roll data set. 2000 points were generated uniformly in a rectangle (top left) and mapped into a Swiss roll configuration in $\mathbf{R}^3$. Ordinary Isomap recovers the rectangular structure correctly provided that the neighbourhood parameter is not too large (in this case $k = 8$ works). The tests show that this peformance is not significantly degraded when L-Isomap is used. For each $n$, we chose $n$ landmark points at random; even down to 4 landmarks the embedding closely approximates the (non-landmark) Isomap embedding. The configuration of three landmarks was chosen especially to illustrate the affine distortion that may arise if the landmarks lie close to a subspace (in this case, a line). For three landmarks chosen at random, results are generally much better.

In contrast, LLE is unstable under changes in its sparseness parameter $k$ (neighbourhood size). To be fair, $k$ is principally a topological parameter and only incidentally a sparseness parameter for LLE. In L-Isomap, these two roles are separately fulfilled by $k$ and $n$.

# 4   Conclusion

Local approaches to nonlinear dimensionality reduction such as LLE or Laplacian Eigenmaps have two principal advantages over a global approach such as Isomap: they tolerate a certain amount of curvature and they lead naturally to a sparse eigenvalue problem. However, neither curvature tolerance nor computational sparsity are explicitly part of the formulation of the local approaches; these features emerge as byproducts of the goal of trying to preserve only the data's local geometric structure. Because they are not explicit goals but only convenient byproducts, they are not in fact reliable features of the local approach. The conformal invariance of LLE can fail in sometimes surprising ways, and the computational sparsity is not tunable independently of the topological sparsity of the manifold. In contrast, we have presented two extensions to Isomap that are explicitly designed to remove a well-characterized form of curvature and to exploit the computational sparsity intrinsic to low-dimensional manifolds. Both extensions are amenable to algorithmic analysis, with provable conditions under which they return accurate results; and they have been tested successfully on challenging data sets.

## Acknowledgments

This work was supported in part by NSF grant DMS-0101364, and grants from Schlumberger, MERL and the DARPA Human ID program. The authors wish to thank Thomas Vetter for providing the range and texture maps for the synthetic face; and Lauren Schmidt for her help in rendering the actual images using Curious Labs' "Poser" software.

## References

[1] Tenenbaum, J.B., de Silva, V. & Langford, J.C (2000) A global geometric framework for nonlinear dimensionality reduction. *Science* **290**: 2319–2323.

[2] Roweis, S. & Saul, L. (2000) Nonlinear dimensionality reduction by locally linear embedding. *Science* **290**: 2323–2326.

[3] Belkin, M. & Niyogi, P. (2002) Laplacian eigenmaps and spectral techniques for embedding and clustering. In T.G. Dietterich, S. Becker and Z. Ghahramani (eds.), *Advances in Neural Information Processing Systems 14*. MIT Press.

[4] Bishop, C., Svensen, M. & Williams, C. (1998) GTM: The generative topographic mapping. *Neural Computation* **10(1)**.

[5] Kohonen, T. (1984) *Self Organisation and Associative Memory.* Springer-Verlag, Berlin.

[6] Bregler, C. & Omohundro, S.M. (1995) Nonlinear image interpolation using manifold learning. In G. Tesauro, D.S. Touretzky & T.K. Leen (eds.), *Advances in Neural Information Processing Systems 7*: 973–980. MIT Press.

[7] DeMers, D. & Cottrell, G. (1993) Non-linear dimensionality reduction In S. Hanson, J. Cowan & L. Giles (eds.), *Advances in Neural Information Processing Systems 5*: 580–590. Morgan-Kaufmann.

[8] Bernstein, M., de Silva, V., Langford, J.C. & Tenenbaum, J.B. (December 2000) Graph approximations to geodesics on embedded manifolds. Preprint may be downloaded at the URL: http://isomap.stanford.edu/BdSLT.pdf

[9] Torgerson, W.S. (1958) *Theory and Methods of Scaling.* Wiley, New York.

[10] Cox, T.F. & Cox M.A.A. (1994) *Multidimensional Scaling.* Chapman & Hall, London.

[11] de Silva, V. & Tenenbaum, J.B. (in preparation) Sparse multidimensional scaling using landmark points.
